# Policy-Gradient Methods for Planning

**Douglas Aberdeen**
Statistical Machine Learning, National ICT Australia, Canberra
`doug.aberdeen@anu.edu.au`

## Abstract

Probabilistic temporal planning attempts to find good policies for acting
in domains with concurrent durative tasks, multiple uncertain outcomes,
and limited resources. These domains are typically modelled as Markov
decision problems and solved using dynamic programming methods.
This paper demonstrates the application of reinforcement learning — in
the form of a policy-gradient method — to these domains. Our emphasis
is large domains that are infeasible for dynamic programming. Our ap-
proach is to construct simple policies, or agents, for each planning task.
The result is a general probabilistic temporal planner, named the Factored
Policy-Gradient Planner (FPG-Planner), which can handle hundreds of
tasks, optimising for probability of success, duration, and resource use.

## 1 Introduction

To date, only a few planning tools have attempted to handle general probabilistic temporal
planning problems. These tools have only been able to produce good policies for relatively
trivial examples. We apply policy-gradient reinforcement learning (RL) to these domains
with the goal of creating tools that produce good policies in real-world domains rather than
perfect policies in toy domains. We achieve this by: (1) factoring the policy into simple
independent policies for starting each task; (2) presenting each policy with critical observa-
tions instead of the entire state; (3) using function approximators for each policy; (4) using
local optimisation methods instead of global optimisation; and (5) using algorithms with
memory requirements that are independent of the state space size.

Policy gradient methods do not enumerate states and are applicable to multi-agent settings
with function approximation [1, 2], thus they are a natural match for our approach to han-
dling large planning problems. We use the GPOMDP algorithm [3] to estimate the gradient
of a long-term average reward of the planner's performance, with respect to the parameters
of each task policy. We show that maximising a simple reward function naturally minimises
plan durations and maximises the probability of reaching the plan goal.

A frequent criticism of policy-gradient methods compared to traditional forward chaining
planners — or even compared to value-based RL methods — is the lack of a clearly inter-
pretable policy. A minor contribution of this paper is a description of how policy-gradient
methods can be used to prune a decision tree over possible policies. After training, the
decision tree can be translated into a list of policy rules.

Previous probabilistic temporal planners include CPTP [4], Prottle [5], Tempastic [6] and a
military operations planner [7]. Most these algorithms use some form of dynamic program-

ming (either RTDP [8] or AO*) to associate values with each state/action pair. However, this requires values to be stored for each encountered state. Even though these algorithms do not enumerate the entire state space their ability to scale is limited by memory size. Even problems with only tens of tasks can produce millions of relevant states. CPTP, Prottle, and Tempastic minimise either plan duration or failure probability, not both. The FPG-Planner minimises both of these metrics and can easily optimise over resources too.

## 2 Probabilistic temporal planning

Tasks are the basic planning unit corresponding to grounded[1] durative actions. Tasks have the effect of setting condition variables to true or false. Each task has a set of preconditions, effects, resource requirements, and a fixed probability of failure. Durations may be fixed or dependent on how long it takes for other conditions to be established. A task is *eligible* to begin when its preconditions are satisfied and sufficient resources are available. A starting task may have some immediate effects. As tasks end a set of effects appropriate to the outcome are applied. Typically, but not necessarily, succeeding tasks set some facts to true, while failing tasks do nothing or negate facts. Resources are occupied during task execution and consumed when the task ends. Different outcomes can consume varying levels of resources. The planning goal is to set a subset of the conditions to a desired value.

The closest work to that presented here is described by Peshkin et al. [1] which describes how a policy-gradient approach can be applied to multi-agent MDPs. This work lays the foundation for this application, but does not consider the planning domain specifically. It is also applied to relatively small domains, where the state space could be enumerated.

Actions in temporal planning consist of launching multiple tasks concurrently. The number of candidate actions available in a given state is the power set of the tasks that are eligible to start. That is, with $N$ eligible tasks there are $2^N$ possible actions. Current planners explore this action space systematically, pruning actions that lead to low rewards. When combined with probabilistic outcomes the state space explosion cripples existing planners for tens of tasks and actions. A key reason treat each task as an individual policy agent is to deal with this explosion of the action space. We replace the single agent choosing from the power-set of eligible tasks with a single simple agent for each task. The policy learnt by each agent is whether to start its associated task given its observation, independent of the decisions made by the other agents. This idea alone does not simplify the problem. Indeed, if the agents received perfect state information they could learn to predict the decision of the other agents and still act optimally. The significant reduction in complexity arises from: (1) restricting the class of functions that represent agents, (2) providing only partial state information, (3) optimising locally, using gradient ascent.

## 3 POMDP formulation of planning

Our intention is to deliberately use simple agents that only consider partial state information. This requires us to explicitly consider partial observability. A finite partially observable Markov decision process consists of: a finite set of states $s \in \mathcal{S}$; a finite set of actions $\mathbf{a} \in \mathcal{A}$; probabilities $\Pr[s'|s, \mathbf{a}]$ of making state transition $s \rightarrow s'$ under action $\mathbf{a}$; a reward for each state $r(s) : \mathcal{S} \rightarrow \mathbb{R}$; and a finite set of observation vectors $\mathbf{o} \in \mathcal{O}$ seen by the agent in place of the complete state descriptions. For this application, observations are drawn deterministically given the state, but more generally may be stochastic. *Goal states* are states where all the goal state variables are satisfied. From *failure states* it is impossible to reach a goal state, usually because time or resources have run out. These two classes of state are combined to form the set of *reset* states that produce an immediate reset to the

initial state $s_0$. A single trajectory through the state space consists of many individual trials that automatically reset to $s_0$ each time a goal state or failure state is reached.

Policies are stochastic, mapping observation vectors **o** to a probability over actions. Let $N$ be the number of basic tasks available to the planner. In our setting an action **a** is a binary vector of length $N$. An entry of 1 at index $n$ means 'Yes' begin task $n$, and a 0 entry means 'No' do not start task $n$. The probability of actions is $\Pr[\mathbf{a}|\mathbf{o}, \theta]$, where conditioning on $\theta$ reflects the fact that the policy is controlled by a set of real valued parameters $\theta \in \mathbb{R}^p$. This paper assumes that all stochastic policies (i.e., any values for $\theta$) reach reset states in finite time when executed from $s_0$. This is enforced by limiting the maximum duration of a plan. This ensures that the underlying MDP is *ergodic*, a necessary condition for GPOMDP. The GPOMDP algorithm maximises the long-term average reward

$$\eta(\theta) = \lim_{T \to \infty} \frac{1}{T} \sum_{t=0}^{T-1} r(s_t).$$

In the context of planning, the instantaneous reward provides the agent with a measure of progress toward the goal. A simple reward scheme is to set $r(s) = 1$ for all states $s$ that represent the goal state, and 0 for all other states. To maximise $\eta(\theta)$, successful planning outcomes must be reached as frequently as possible. This has the desired property of simultaneously minimising plan duration, as well as maximising the probability of reaching the goal (failure states achieve no reward). It is tempting to provide a negative reward for failure states, but this can introduce poor local maxima in the form of policies that avoid negative rewards by avoiding progress altogether. We provide a reward of 1000 each time the goal is achieved, plus an admissible heuristic reward for progress toward the goal. This additional *shaping* reward provides a reward of 1 for every goal state variable achieved, and -1 for every goal variable that becomes unset. Policies that are optimal with the additional shaping reward are still optimal under the basic goal state reward [9].

### 3.1 Planning state space

For probabilistic temporal planning our state description contains [7]: the state's absolute time, a queue of impending events, the status of each task, the truth value of each condition, and the available resources. In a particular state, only a subset of the eligible tasks will satisfy all preconditions for execution. We call these tasks *eligible*. When a decision to start a fixed duration task is made an end-task event is added to a time ordered event queue. The event queue holds a list of events that the planner is committed to, although the outcome of those events may be uncertain.

The generation of successor states is shown in Alg. 1. The algorithm begins by starting the tasks given by the current action, implementing any immediate effects. An end-task event is added at an appropriate time in the queue. The state update then proceeds to process events until there is at least one task that is eligible to begin. Events have probabilistic outcomes. Line 20 of Alg. 1 samples one possible outcome from the distribution imposed by probabilities in the problem definition. Future states are only generated at points where tasks can be started. Thus, if an event outcome is processed and no tasks are enabled, the search recurses to the next event in the queue.

## 4 Factored Policy-Gradient

We assume the presence of policy agents, parameterised with independent sets of parameters for each agent $\theta = \{\theta_1, \ldots, \theta_N\}$. We seek to adjust the parameters of the policy to maximise the long-term average reward $\eta(\theta)$. The GPOMDP algorithm [3] estimates the gradient $\nabla \eta(\theta)$ of the long-term average reward with respect to the current set of policy

Alg. 1: **findSuccessor(State $s$, Action $\mathbf{a}$)**
1: **for** each $a_n$ ='Yes' in $\mathbf{a}$ **do**
2:     $s$.beginTask($n$)
3:     $s$.addEvent($n$, $s$.time+taskDuration($n$))
4: **end for**
5: **repeat**
6:     **if** $s$.time > maximum makespan **then**
7:         $s$.failureLeaf=true
8:         return
9:     **end if**
10:     **if** $s$.operationGoalsMet() **then**
11:         $s$.goalLeaf=true
12:         return
13:     **end if**
14:     **if** $\neg s$.anyEligibleTasks() **then**
15:         $s$.failureLeaf=true
16:         return
17:     **end if**
18:     $event = s$.nextEvent()
19:     $s$.time $= event$.time
20:     sample $outcome$ from $event$
21:     $s$.implementEffects($outcome$)
22: **until** $s$.anyEligibleTasks()

Alg. 2: **Gradient Estimator**
1: Set $s_0$ to initial state, $t = 0$, $\mathbf{e}_t = [0]$
2: **while** $t < T$ **do**
3:     $\mathbf{e}_t = \beta \mathbf{e}_{t-1}$
4:     Generate observation $\mathbf{o}_t$ of $s_t$
5:     **for** Each eligible task $n$ **do**
6:         Sample $a_{tn}$ =Yes or $a_{tn}$ =No
7:         $\mathbf{e}_t = \mathbf{e}_t + \nabla \log \Pr[a_{tn}|\mathbf{o}, \theta_n]$
8:     **end for**
9:     Try action $\mathbf{a}_t = \{a_{t1}, a_{t2}, \ldots, a_{tN}\}$
10:     **while** mutex prohibits $\mathbf{a}_t$ **do**
11:         randomly disable task in $\mathbf{a}_t$
12:     **end while**
13:     $s_{t+1} = \text{findSuccessor}(s_t, \mathbf{a}_t)$
14:     $\hat{\nabla}_t \eta(\theta) = \hat{\nabla}_{t-1} \eta(\theta) - \frac{1}{t+1}(r(s_{t+1})\mathbf{e}_t - \hat{\nabla}_{t-1}\eta(\theta))$
15:     $t \leftarrow t + 1$
16: **end while**
17: Return $\hat{\nabla}_T \eta(\theta)$

parameters. Once an estimate $\hat{\nabla}\eta(\theta)$ is computed over $T$ simulation steps, we maximise the long-term average reward with the gradient ascent $\theta \leftarrow \theta + \alpha\hat{\nabla}\eta(\theta)$, where $\alpha$ is a small step size. The experiments in this paper use a line search to determine good values of $\alpha$. We do not guarantee that the best representable policy is found, but our experiments have produced policies comparable to global methods like real-time dynamic programming [8].

The algorithm works by sampling a single long trajectory through the state space (Fig. 4): (1) the first state represents time 0 in the plan; (2) the agents all receive the vector observation $\mathbf{o}_t$ of the current state $s_t$; (3) each agent representing an eligible task emits a probability of starting; (4) each agent samples start or do not start and issues it as a planning action; (5) the state transition is sampled with Alg. 1; (6) the agents receive the global reward for the new state action and update their gradient estimates. Steps 1 to 6 are repeated $T$ times.

Each vector action $\mathbf{a}_t$ is a combination of independent 'Yes' or 'No' choices made by each eligible agent. Each agent is parameterised by an independent set of parameters that make up $\theta \in \mathbb{R}^p$: $\theta_1, \theta_2, \ldots, \theta_N$. If $a_{tn}$ represents the binary decision made by agent $n$ at time $t$ about whether to start its corresponding task, then the policy factors into

$$\Pr[\mathbf{a}_t|\mathbf{o}_t, \theta] = \Pr[a_{t1}, \ldots, a_{tN}|\mathbf{o}_t, \theta_1, \ldots, \theta_N]$$
$$= \Pr[a_{t1}|\mathbf{o}_t, \theta_1] \times \cdots \times \Pr[a_{tN}|\mathbf{o}_t, \theta_N].$$

It is not necessary for all agents to receive the same observation, and it may be advantageous to show different agents different parts of the state, leading to a *decentralised* planning algorithm. Similar approaches are adopted by Peshkin et al. [1], Tao et al. [2], using policy-gradient methods to train multi-agent systems. The main requirement for each policy-agent is that $\log \Pr[a_{tn}|\mathbf{o}_t, \theta_n]$ be differentiable with respect to the parameters for each choice task start $a_{tn}$ ='Yes' or 'No'. We now describe two such agents.

### 4.1 Linear approximator agents

One representation of agents is a linear network mapped into probabilities using a logistic regression function:

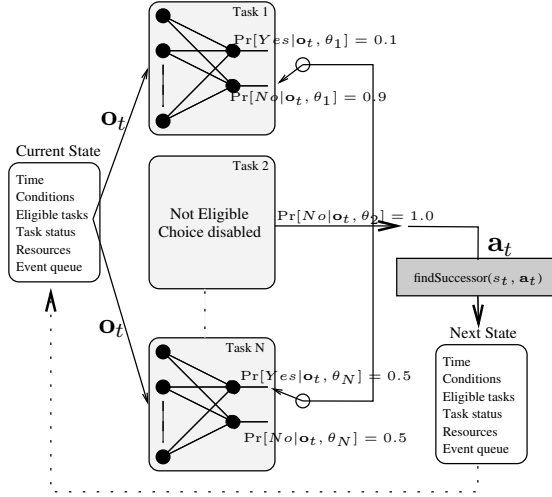

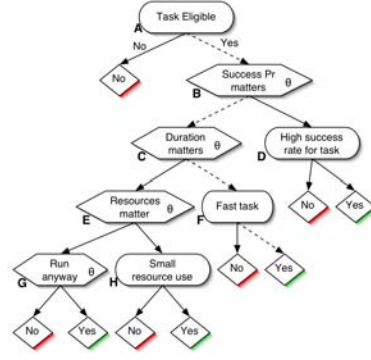

Fig. 3: Decision tree agent.

Fig. 4: (Left) Individual task-policies make independent decisions.

$$\Pr[a_{tn} = Yes | \mathbf{o}_t, \theta_n] = \frac{exp(\mathbf{o}_t^\top \theta_n)}{exp(\mathbf{o}_t^\top \theta_n) + 1} \tag{1}$$

If the dimension of the observation vector is $|\mathbf{o}|$ then each set of parameters $\theta_n$ can be thought of as an $|\mathbf{o}|$ vector that represents the approximator weights for task $n$. The log derivatives, necessary for Alg. 2, are given in [10]. Initially, the parameters are set to small random values: a near uniform random policy. This encourages exploration of the action space. Each gradient step typically moves the parameters closer to a deterministic policy. After some experimentation we chose an observation vector that is a binary description of the eligible tasks and the state variable truth values plus a constant 1 bit to provide bias to the agents' linear networks.

## 4.2 Decision tree agents

Often we have a selection of potential control rules. A decision tree can represent all such control rules at the leaves. The nodes are additional parameterised or hardwired rules that select between different branches, and therefore different control rules. An action $a$ is selected by starting at the root node and following a path down the tree, visiting a set of decision nodes $\mathcal{D}$. At each node we either applying a hard coded branch selection rule, or sample a stochastic branch rule from the probability distribution invoked by the parameterisation. Assuming the independence of decisions at each node, the probability or reaching an action leaf $l$ equals the product of branch probabilities at each decision node

$$\Pr[a = l | \mathbf{o}, \theta] = \prod_{d \in \mathcal{D}} \Pr[d' | \mathbf{o}, \theta_d], \tag{2}$$

where $d$ represents the current decision node, and $d'$ represents the next node visited in the tree. The final next node $d'$ is the leaf $l$. The probability of a branch followed as a result of a hard-coded rule is 1. The individual $\Pr(d' | \mathbf{o}, \theta_d)$ functions can be any differentiable function of the observation vector $\mathbf{o}$.

For multi-agent domains, such as our formulation of planning, we have a decision tree for each task agent. We use the same initial tree (with different parameters), for each agent, shown in Fig. 3. Nodes A, D, F, H represent hard coded rules that switch with probability one between the Yes and No branches based on a boolean observation that gives the truth of the statement in the node for the current state. Nodes B, C, E, G are parameterised so

that they select branches stochastically. For this application, the probability of choosing the Yes or No branches is a single parameter logistic function that is independent of the observations. Parameter adjustments have the simple effect of pruning parts the tree that represent poor policies, leaving the hard coded rules to choose the best action given the observation. The policy encoded by the parameter is written in the node label. For example for task agent $n$, and decision node C "task duration matters?", we have the probability

$$\Pr(Yes|\mathbf{o}, \theta_{n,C}) = \Pr(Yes|\theta_{n,C}) = \frac{exp(\theta_{n,C})}{exp(\theta_{n,C}) + 1}$$

The log gradient of this function is given in [10]. If set parameters to always select the dashed branch in Fig. 3 we would be following the policy: *if the task IS eligible, and probability this task success does NOT matter, and the duration of this task DOES matter, and this task IS fast, then start, otherwise do not start.* Apart from being easy to interpret the optimised decision tree as a set of — possibly stochastic — if-then rules, we can also encode highly expressive policies with only a few parameters.

### 4.3 GPOMDP for planning

Alg. 4 describes the algorithm for computing $\hat{\nabla}\eta(\theta)$, based on GPOMDP [3]. The vector quantity $\mathbf{e}_t$ is an eligibility trace. It has dimension $p$ (the total number of parameters), and can be thought of as storing the eligibility of each parameter for being reinforced after receiving a reward. The gradient estimate provably converges to a biased estimate of $\nabla\eta(\theta)$ as $T \to \infty$. The quantity $\beta \in [0, 1)$ controls the degree of bias in the estimate. As $\beta$ approaches 1, the bias of the estimates drop to 0. However if $\beta = 1$, estimates exhibit infinite variance in the limit as $T \to \infty$. Thus the parameter $\beta$ is used to achieve a bias/variance tradeoff in our stochastic gradient estimates. GPOMDP gradient estimates have been proven to converge, even under partial observability.

Line 8 computes the log gradient of the sampled action probability and adds the gradient for the $n$'th agent's parameters into the eligibility trace. The gradient for parameters not relating to agent $n$ is 0. We do not compute $\Pr[a_{tn}|\mathbf{o}_t, \theta_n]$ or gradients for tasks with unsatisfied preconditions. If all eligible agents decide *not* to start their tasks, we issue a null-action. If the state event queue is not empty, we process the next event, otherwise time is incremented by 1 to ensure all possible policies will eventually reach a reset state.

## 5 Experiments

### 5.1 Comparison with previous work

We compare the FPG-Planner with that of our earlier RTDP based planner for military operations [7], which is based on real-time dynamic programming with [8]. The domains come from the Australian Defence Science and Technology Organisation, and represent military operations planning scenarios. There are two problems, the first with 18 tasks and 12 conditions, and the second with 41 tasks and 51 conditions. The goal is to set the "Objective island secured" variable to true. There are multiple interrelated tasks that can lead to the goal state. Tasks fail or succeed with a known probability and can only execute once, leading to relatively large probabilities of failure even for optimal plans. See [7] for details. Unless stated, FPG-Planner experiments used $T = 500,000$ gradient estimation steps and $\beta = 0.9$. Optimisation time was limited to 20 minutes wall clock time on a single user 3GHz Pentium IV with 1GB ram. All evaluations are based on 10,000 simulated executions of finalised policies. Results quote the average duration, resource consumption, and the percentage of plans that terminate in a failure state.

We repeat the comparison experiments 50 times with different random seeds and report

Table 1: Two domains compared with a dynamic programming based planner.

| Problem | RTDP | | | Factored Linear | | | Factored Tree | | |
|---|---|---|---|---|---|---|---|---|---|
| | Dur | Res | Fail% | Dur | Res | Fail% | Dur | Res | Fail% |
| Assault Ave | 171 | 8.0 | 26.1 | 105 | 8.3 | 26.6 | 115 | 8.3 | 27.1 |
| Assault Best | 113 | 6.2 | 24.0 | 93.1 | 8.7 | 23.1 | 112 | 8.4 | 25.6 |
| Webber Ave | 245 | 4.4 | 58.1 | 193 | 4.1 | 57.9 | 186 | 4.1 | 58.0 |
| Webber Best | 217 | 4.2 | 57.7 | 190 | 4.1 | 57.0 | 181 | 4.1 | 57.3 |

Table 3: Results for the Art45/25 domain.

Table 2: Effect of different observations.

| Observation | Dur | Res | Fail% |
|---|---|---|---|
| Eligible & Conds | 105 | 8.3 | 26.6 |
| Conds only | 112 | 8.1 | 28.1 |
| Eligible only | 112 | 8.1 | 29.6 |

| Policy | Dur | Res | Fail% |
|---|---|---|---|
| Random | 394 | 206 | 83.4 |
| Naive | 332 | 231 | 78.6 |
| Linear | 121 | 67 | 7.4 |
| Dumb Tree | 157 | 92 | 19.1 |
| Prob Tree | 156 | 62 | 10.9 |
| Dur Tree | 167 | 72 | 17.4 |
| Res Tree | 136 | 53 | 8.50 |

mean and best results in Table 1. The "Best" plan minimises an arbitrarily chosen combined metric of $10 \times fail\% + dur$. FPG-Planning with a linear approximator significantly shortens the duration of plans, without increasing the failure rate. The very simple decision tree performs less well than than the linear approximator, but better than the dynamic programming algorithm. This is somewhat surprising given the simplicity of the tree for each task. The shorter duration for the Webber decision tree is probably due to the slightly higher failure rate. Plans failing early produces shorter durations.

Table 1 assumes that the observation vector **o** presented to linear agents is a binary description of the eligible tasks and the condition truth values plus a constant 1 bit to provide bias to the agents' linear networks. Table 2 shows that giving the agents less information in the observation harms performance.

## 5.2 Large artificial domains

Each scenario consists of $N$ tasks and $C$ state variables. The goal state of the synthetic scenarios is to assert 90% of the state variables, chosen during scenario synthesis, to be true. See [10] for details. All generated problems have scope for choosing tasks instead of merely scheduling them. All synthetic scenarios are guaranteed to have at least one policy which will reach the operation goal assuming all tasks succeed. Even a few tens of tasks and conditions can generate a state space too large for main memory.

We generated 37 problems, each with 40 tasks and 25 conditions (Art40/25). Although the number of tasks and conditions is similar to the Webber problem described above, these problems demonstrate significantly more choices to the planner, making planning nontrivial. Unlike the initial experiments, all tasks can be repeated as often as necessary so the overall probability of failure depends on how well the planner chooses and orders tasks to avoid running out of time and resources. Our RTDP based planner was not able to perform any significant optimisation in 20m due to memory problems. Thus, to demonstrate FPG-Planning is having some effect, we compared the optimised policies to two simple policies. The *random* policy starts each eligible task with probability 0.5. The *naive* policy starts all eligible tasks. Both of these policies suffer from excessive resource consumption and negative effects that can cause failure.

Table 3 shows that the linear approximator produces the best plans, but it requires $C + 1$ parameters per task. The results for the decision tree illustrated in Fig. 3 are given in the

"Prob Tree" row. This tree uses a constant 4 parameters per task, and subsequently requires fewer operations when computing gradients. The "Dumb" row is a decision stub, with one parameter per task that simply learns whether to start when eligible. The remaining "Dur" and "Res" Tree rows re-order the nodes in Fig. 3 to swap the nodes C and E respectively with node B. This tests the sensitivity of the tree to node ordering. There appears to be significant variation in the results. For example, when node E is swapped with B, the resultant policies use less resources.

We also performed optimisation of a 200 task, 100 condition problem generated using the same rules as the Art40/25 domain. The naive policy had a failure rate of 72.4%. No time limit was applied. Linear network agents (20,200 parameters) optimised for 14 hours, before terminating with small gradients, and resulted in a plan with 20.8% failure rate. The decision tree agent (800 parameters) optimised for 6 hours before terminating with a 1.7% failure rate. The smaller number of parameters and a priori policies embedded in the tree, allow the decision tree to perform well in very large domains. Inspection of the resulting parameters demonstrated that different tasks pruned different regions of the decision tree.

## 6 Conclusion

We have demonstrated an algorithm with great potential to produce good policies in real-world domains. Further work will refine our parameterised agents, and validate this approach on realistic larger domains. We also wish to characterise possible local minima.

**Acknowledgements**

Thank you to Olivier Buffet and Sylvie Thiébaux for many helpful comments. National ICT Australia is funded by the Australian Government's Backing Australia's Ability program and the Centre of Excellence program. This project was also funded by the Australian Defence Science and Technology Organisation.

**References**

[1] L. Peshkin, K.-E. Kim, N. Meuleau, and L. P. Kaelbling. Learning to cooperate via policy search. In *UAI*, 2000.

[2] Nigel Tao, Jonathan Baxter, and Lex Weaver. A multi-agent, policy-gradient approach to network routing. In *Proc. ICML'01*. Morgan Kaufmann, 2001.

[3] J. Baxter, P. Bartlett, and L. Weaver. Experiments with infinite-horizon, policy-gradient estimation. *JAIR*, 15:351–381, 2001.

[4] Mausam and Daniel S. Weld. Concurrent probabilistic temporal planning. In *Proc. International Conference on Automated Planning and Scheduling*, Moneteray, CA, June 2005. AAAI.

[5] I. Little, D. Aberdeen, and S. Thiébaux. Prottle: A probabilistic temporal planner. In *Proc. AAAI'05*, 2005.

[6] Hakan L. S. Younes and Reid G. Simmons. Policy generation for continuous-time stochastic domains with concurrency. In *Proc. of ICAPS'04*, volume 14, 2005.

[7] Douglas Aberdeen, Sylvie Thiébaux, and Lin Zhang. Decision-theoretic military operations planning. In *Proc. ICAPS*, volume 14, pages 402–411. AAAI, June 2004.

[8] A.G. Barto, S. Bradtke, and S. Singh. Learning to act using real-time dynamic programming. *Artificial Intelligence*, 72, 1995.

[9] A.Y. Ng, D. Harada, and S. Russell. Policy invariance under reward transformations: Theory and application to reward shaping. In *Proc. ICML'99*, 1999.

[10] Douglas Aberdeen. The factored policy-gradient planner. Technical report, NICTA, 2005.

## Footnotes

[1]Grounded means that tasks do not have parameters that can be instantiated.
